# Sparse Kernel Orthonormalized PLS for feature extraction in large data sets

**Jerónimo Arenas-García, Kaare Brandt Petersen and Lars Kai Hansen**
Informatics and Mathematical Modelling
Technical University of Denmark
DK-2800 Kongens Lyngby, Denmark
`{jag,kbp,lkh}@imm.dtu.dk`

## Abstract

In this paper we are presenting a novel multivariate analysis method. Our scheme is based on a novel kernel orthonormalized partial least squares (PLS) variant for feature extraction, imposing sparsity constrains in the solution to improve scalability. The algorithm is tested on a benchmark of UCI data sets, and on the analysis of integrated short-time music features for genre prediction. The upshot is that the method has strong expressive power even with rather few features, is clearly outperforming the ordinary kernel PLS, and therefore is an appealing method for feature extraction of labelled data.

## 1 Introduction

Partial Least Squares (PLS) is, in its general form, a family of techniques for analyzing relations between data sets by latent variables. It is a basic assumption that the information is overrepresented in the data sets, and that these therefore can be reduced in dimensionality by the latent variables. Exactly how these are found and how the data is projected varies within the approach, but they are often maximizing the covariance of two projected expressions. One of the appealing properties of PLS, which has made it popular, is that it can handle data sets with more dimensions than samples and massive collinearity between the variables.

The basic PLS algorithm considers two data sets $\mathbf{X}$ and $\mathbf{Y}$, where samples are arranged in rows, and consists on finding latent variables which account for the covariance $\mathbf{X}^T\mathbf{Y}$ between the data sets. This is done either as an iterative procedure or as an eigenvalue problem. Given the latent variables, the data sets $\mathbf{X}$ and $\mathbf{Y}$ are then transformed in a process which subtracts the information contained in the latent variables. This process, which is often referred to as *deflation*, can be done in a number of ways and these different approaches are defining the many variants of PLS.

Among the many variants of PLS, the one that has become particularly popular is the algorithm presented in [17] and studied in further details in [3]. The algorithm described in these, will in this paper be referred to as PLS2, and is based on the following two assumptions: First, that the latent variables of $\mathbf{X}$ are good predictors of $\mathbf{Y}$ and, second, that there is a linear relation between the latent variables of $\mathbf{X}$ and of $\mathbf{Y}$. This linear relation is implying a certain deflation scheme, where the latent variable of $\mathbf{X}$ is used to deflate also the $\mathbf{Y}$ data set. Several other variants of PLS exist such as "PLS Mode A" [16], Orthonormalized PLS [18] and PLS-SB [11]; see [1] for a discussion of the early history of PLS, [15] for a more recent and technical description, and [9] and for a very well-written contemporary overview.

No matter how refined the various early developments of PLS become, they are still linear projections. Therefore, in the cases where the variables of the input and output spaces are not linearly related, the challenge of the data is still poorly handled. To counter this, different non-linear versions of PLS have been developed, and these can be categorized on two fundamentally different

approaches: 1) The modified PLS2 variants in which the linear relation between the latent variables is substituted by a non-linear relation; and 2) the kernel variants in which the PLS algorithm has been reformulated to fit a kernel approach. In the second approach, the input data is mapped by a non-linear function into a high-dimensional space in which ordinary linear PLS is performed on the transformed data. A central property of this kernel approach is, as always, the exploitation of the kernel trick, i.e., that only the inner products in the transformed space are necessary and not the explicit non-linear mapping. It was Rosipal and Trejo who first presented a non-linear kernel variant of PLS in [7]. In that paper, the kernel matrix and the $\mathbf{Y}$ matrix are deflated in the same way, and the PLS variant is thus more in line with the PLS2 variant than with the traditional algorithm from 1975 (PLS Mode A). The non-linear kernel PLS by Rosipal and Trejo is in this paper referred to as simply KPLS2, although many details could advocate more detailed nomenclator.

The appealing property of kernel algorithms in general is that one can obtain the flexibility of non-linear expressions while still solving only linear equations. The downside is that for a data set of $l$ samples, the kernel matrices to be handled are $l \times l$, which, even for a moderate number of samples, quickly becomes a problem with respect to both memory and computing time. This problem is present not only in the training phase, but also when predicting the output given some large training data set: evaluating thousands of kernels for every new input vector is, in most applications, not acceptable. Furthermore, there is, for these so-called dense solutions in multivariate analysis, also the problem of overfitting. To counter the impractical dense solutions in kernel PLS, a few solutions have been proposed: In [2], the feature mapping directly is approximated following the Nystrom method, and in [6] the underlying cost function is modified to impose sparsity.

In this paper, we introduce a novel kernel PLS variant called Reduced Kernel Orthonormalized Partial Least Squares (rKOPLS) for large scale feature extraction. It consists of two parts: A novel orthonormalized variant of kernel PLS called KOPLS, and a sparse approximation for large scale data sets. Compared to related approaches like [8], the KOPLS is transforming only the input data, and is keeping them orthonormal at two stages: the images in feature space and the projections in feature space. The sparse approximation is along the lines of [4], that is, we are representing the reduced kernel matrix as an outer product of a reduced and a full feature mapping, and thus keeping more information than changing the cost function or doing simple subsampling.

Since rKOPLS is specially designed to handle large data sets, our experimental work will focus on such data sets, paying extra attention to the prediction of music genre, an application that typically involves large amount of high dimensional data. The abilities of our algorithm to discover non-linear relations between input and output data will be illustrated, as will be the relevance of the derived features compared to those provided by an existing kernel PLS method.

The paper is structured as follows: In Section 2, the novel kernel orthonormalized PLS variant is introduced, and in Section 3 the sparse approximation is presented. Section 4 shows numerical results on UCI benchmark data sets, and on the above mentioned music application. In the last section, the main results are summarized and discussed.

## 2 Kernel Orthonormalized Partial Least Squares

Consider we are given a set of pairs $\{\phi(\mathbf{x}_i), \mathbf{y}_i\}_{i=1}^l$, with $\mathbf{x}_i \in \Re^N$, $\mathbf{y}_i \in \Re^M$, and $\phi(\mathbf{x}) : \Re^N \rightarrow \mathcal{F}$ a function that maps the input data into some Reproducing Kernel Hilbert Space (RKHS), usually referred to as feature space, of very large or even infinite dimension. Let us also introduce the matrices $\boldsymbol{\Phi} = [\phi(\mathbf{x}_1), \ldots, \phi(\mathbf{x}_l)]^T$ and $\mathbf{Y} = [\mathbf{y}_1, \ldots, \mathbf{y}_l]^T$, and denote by

$$\boldsymbol{\Phi}' = \boldsymbol{\Phi}\mathbf{U} \quad \text{and} \quad \mathbf{Y}' = \mathbf{Y}\mathbf{V}$$

two matrices, each one containing $n_p$ projections of the original input and output data, $\mathbf{U}$ and $\mathbf{V}$ being the projection matrices of sizes $dim(\mathcal{F}) \times n_p$ and $M \times n_p$, respectively. The objective of (kernel) Multivariate Analysis (MVA) algorithms is to search for projection matrices such that the projected input and output data are maximally aligned. For instance, Kernel Canonical Correlation Analysis (KCCA) finds the projections that maximize the correlation between the projected data, while Kernel Partial Least Squares (KPLS) provides the directions for maximum covariance:

$$
\begin{aligned}
\text{KPLS}: \quad &\text{maximize:} \quad \text{Tr}\{\mathbf{U}^T \tilde{\boldsymbol{\Phi}}^T \tilde{\mathbf{Y}}\mathbf{V}\} \\
&\text{subject to:} \quad \mathbf{U}^T\mathbf{U} = \mathbf{V}^T\mathbf{V} = \mathbf{I}
\end{aligned}
\tag{1}
$$

where $\tilde{\mathbf{\Phi}}$ and $\tilde{\mathbf{Y}}$ are centered versions of $\mathbf{\Phi}$ and $\mathbf{Y}$, respectively, $\mathbf{I}$ is the identity matrix of size $n_p$, and the $T$ superscript denotes matrix or vector transposition. In this paper, we propose a kernel extension of a different MVA method, namely, the Orthonormalized Partial Least Squares [18]. Our proposed kernel variant, called KOPLS, can be stated in the kernel framework as

$$\text{KOPLS}: \quad \begin{aligned} &\text{maximize:} \quad \text{Tr}\{\mathbf{U}^T\tilde{\mathbf{\Phi}}^T\tilde{\mathbf{Y}}\tilde{\mathbf{Y}}^T\tilde{\mathbf{\Phi}}\mathbf{U}\} \\ &\text{subject to:} \quad \mathbf{U}^T\tilde{\mathbf{\Phi}}^T\tilde{\mathbf{\Phi}}\mathbf{U} = \mathbf{I} \end{aligned} \quad (2)$$

Note that, unlike KCCA or KPLS, KOPLS only extracts projections of the input data. It is known that Orthonormalized PLS is optimal for performing linear regression on the input data when a bottleneck is imposed for data dimensionality reduction [10]. Similarly, KOPLS provides optimal projections for linear multi-regression in feature space. In other words, the solution to (2) also minimizes the sum of squares of the residuals of the approximation of the label matrix:

$$\|\tilde{\mathbf{Y}} - \tilde{\mathbf{\Phi}}'\hat{\mathbf{B}}\|_F^2, \qquad \hat{\mathbf{B}} = (\tilde{\mathbf{\Phi}}'^T\tilde{\mathbf{\Phi}}')^{-1}\tilde{\mathbf{\Phi}}'^T\tilde{\mathbf{Y}} \qquad (3)$$

where $\|\cdot\|_F$ denotes the Frobenius norm of a matrix and $\hat{\mathbf{B}}$ is the optimal regression matrix. Similarly to other MVA methods, KOPLS is not only useful for multi-regression problems, but it can also be used as a very powerful kernel feature extractor in supervised problems, including also the multi-label case, when $\mathbf{Y}$ is used to encode class membership information. The optimality condition suggests that the features obtained by KOPLS will be more relevant than those provided by other MVA methods, in the sense that they will allow similar or better accuracy rates using fewer projections, a conjecture that we will investigate in the experiments section of the paper.

Coming back to the KOPLS optimization problem, when projecting data into an infinite dimensional space, we need to use the Representer Theorem that states that each of the projection vectors in $\mathbf{U}$ can be expressed as a linear combination of the training data. Then, introducing $\mathbf{U} = \tilde{\mathbf{\Phi}}^T\mathbf{A}$ into (2), where $\mathbf{A} = [\boldsymbol{\alpha}_1, \dots, \boldsymbol{\alpha}_{n_p}]$ and $\boldsymbol{\alpha}_i$ is an $l$-length column vector containing the coefficients for the $i$th projection vector, the maximization problem can be reformulated as:

$$\begin{aligned} &\text{maximize:} \quad \text{Tr}\{\mathbf{A}^T\mathbf{K}_x\mathbf{K}_y\mathbf{K}_x\mathbf{A}\} \\ &\text{subject to:} \quad \mathbf{A}^T\mathbf{K}_x\mathbf{K}_x\mathbf{A} = \mathbf{I} \end{aligned} \qquad (4)$$

where we have defined the centered kernel matrices $\mathbf{K}_x = \tilde{\mathbf{\Phi}}\tilde{\mathbf{\Phi}}^T$ and $\mathbf{K}_y = \tilde{\mathbf{Y}}\tilde{\mathbf{Y}}^T$, such that only inner products in $\mathcal{F}$ are involved [1]. Applying ordinary linear algebra to (4), it can be shown that the columns of $\mathbf{A}$ are given by the solutions to the following generalized eigenvalue problem:

$$\mathbf{K}_x\mathbf{K}_y\mathbf{K}_x\boldsymbol{\alpha} = \lambda\mathbf{K}_x\mathbf{K}_x\boldsymbol{\alpha} \qquad (5)$$

There are a number of ways to solve the above problem. We propose a procedure consisting of iteratively calculating the best projection vector, and then deflating the involved matrices. In short, the optimization procedure at step $i$ consists of the following two differentiated stages:

1. Find the largest generalized eigenvalue of (5), and its corresponding generalized eigenvector: $\{\lambda_i, \boldsymbol{\alpha}_i\}$. Normalize $\boldsymbol{\alpha}_i$ so that condition $\boldsymbol{\alpha}_i\mathbf{K}_x\mathbf{K}_x\boldsymbol{\alpha}_i = 1$ is satisfied.

2. Deflate the $l \times l$ matrix $\mathbf{K}_x\mathbf{K}_y\mathbf{K}_x$ according to:

$$\mathbf{K}_x\mathbf{K}_y\mathbf{K}_x \leftarrow \mathbf{K}_x\mathbf{K}_y\mathbf{K}_x - \lambda_i\mathbf{K}_x\mathbf{K}_x\boldsymbol{\alpha}_i\boldsymbol{\alpha}_i^T\mathbf{K}_x\mathbf{K}_x$$

   The motivation for this deflation strategy can be found in [13], in the discussion of generalized eigenvalue problems. Some intuition can be obtained if we observe its equivalence with

$$\mathbf{K}_y \leftarrow \mathbf{K}_y - \lambda_i\mathbf{K}_x\boldsymbol{\alpha}_i\boldsymbol{\alpha}_i^T\mathbf{K}_x$$

   which accounts for removing from the label matrix $\mathbf{Y}$ the best approximation based on the projections computed at step $i$, i.e., $\mathbf{K}_x\boldsymbol{\alpha}_i$. It can be shown that this deflation scheme decreases by 1 the rank of $\mathbf{K}_x\mathbf{K}_y\mathbf{K}_x$ at each step. Since the rank of the original matrix $\mathbf{K}_y$ is at most $rank(\mathbf{Y})$, this is the maximum number of projections that can be derived when using KOPLS.

This iterative algorithm, which is very similar in nature to the iterative algorithms used for other MVA approaches, has the advantage that, at every iteration, the achieved solution is optimal with respect to the current number of projections.

# 3 Compact approximation of the KOPLS solution

The kernel formulation of the OPLS algorithm we have just presented suffers some drawbacks. In particular, as most other kernel methods, KOPLS requires the computation and storage of a kernel matrix of size $l \times l$, which limits the maximum size of the datasets where the algorithm can be applied. In addition to this, algebraic procedures to solve the generalized eigenvalue problem (5) normally require the inversion of matrix $\mathbf{K}_x \mathbf{K}_x$ which is usually rank deficient. Finally, the matrix $\mathbf{A}$ will in general be dense rather than sparse, a fact which implies that when new data needs to be projected, it will be necessary to compute the kernels between the new data and all the samples in the training data set.

Although it is possible to think of different solutions for each of the above issues, our proposal here is to impose sparsity in the projection vectors representation, i.e., we will use the approximation $\mathbf{U} = \mathbf{\Phi}_R^T \mathbf{B}$, where $\mathbf{\Phi}_R$ is a subset of the training data containing only $R$ patterns ($R < l$) and $\mathbf{B} = [\boldsymbol{\beta}_1, \cdots, \boldsymbol{\beta}_{n_p}]$ contains the parameters of the compact model. Although more sophisticated strategies can be followed in order to select the training data to be incorporated into the basis $\mathbf{\Phi}_R$, we will rely on random selection, very much in the line of the sparse greedy approximation proposed in [4] to reduce the computational burden of Support Vector Machines (SVMs).

Replacing $\mathbf{U}$ in (2) by its approximation, we get an alternative maximization problem that constitutes the basis for a KOPLS algorithm with reduced complexity (rKOPLS):

$$\text{rKOPLS}: \quad \begin{aligned} &\text{maximize:} \quad \text{Tr}\{\mathbf{B}^T \mathbf{K}_R \mathbf{K}_y \mathbf{K}_R^T \mathbf{B}\} \\ &\text{subject to:} \quad \mathbf{B}^T \mathbf{K}_R \mathbf{K}_R^T \mathbf{B} = \mathbf{I} \end{aligned} \tag{6}$$

where we have defined $\mathbf{K}_R = \mathbf{\Phi}_R \tilde{\mathbf{\Phi}}^T$, which is a reduced kernel matrix of size $R \times l$. Note that, to keep the algorithm as simple as possible, we decided not to center the patterns in the basis $\mathbf{\Phi}_R$. Our simulation results suggest that centering $\mathbf{\Phi}_R$ does not result in improved performance. Similarly to the standard KOPLS algorithm, the projections for the rKOPLS algorithm can be obtained by solving

$$\mathbf{K}_R \mathbf{K}_y \mathbf{K}_R^T \boldsymbol{\beta} = \lambda \mathbf{K}_R \mathbf{K}_R^T \boldsymbol{\beta} \tag{7}$$

The iterative two-stage procedure described at the end of the previous section can still be used by simple replacement of the following matrices and variables:

| KOPLS | rKOPLS |
|---|---|
| $\boldsymbol{\alpha}_i$ | $\boldsymbol{\beta}_i$ |
| $\mathbf{K}_x \mathbf{K}_x$ | $\mathbf{K}_R \mathbf{K}_R^T$ |
| $\mathbf{K}_x \mathbf{K}_y \mathbf{K}_x$ | $\mathbf{K}_R \mathbf{K}_y \mathbf{K}_R^T$ |

To conclude the presentation of the rKOPLS algorithm, let us summarize some of its more relevant properties, and how it solves the different limitations of the standard KOPLS formulation:

- Unlike KOPLS, the solution provided by rKOPLS is enforced to be sparse, so that new data is projected with only $R$ kernel evaluations per pattern (in contrast to $l$ evaluations for KOPLS). This is a very desirable property, specially when dealing with large data sets.

- Training rKOPLS projections only requires the computation of a reduced kernel matrix $\mathbf{K}_R$ of size $R \times l$. Nevertheless, note that the approach we have followed is very different to subsampling, since rKOPLS is still using all training data in the MVA objective function.

- The rKOPLS algorithm only needs matrices $\mathbf{K}_R \mathbf{K}_R^T$ and $\mathbf{K}_R \mathbf{K}_y \mathbf{K}_R^T$. It is easy to show that both matrices can be calculated without explicitly computing $\mathbf{K}_R$, so that memory requirements go down to $O(R^2)$ and $O(RM)$, respectively. Again, this is a very convenient property when dealing with large scale problems.

- Parameter $R$ acts as a sort of regularizer, making $\mathbf{K}_R \mathbf{K}_R^T$ full rank.

Table 1 compares the complexity of KOPLS and rKOPLS, as well as that of the KPLS2 algorithm. Note that KPLS2 does not admit a compact formulation as the one we have used for the new method, since the full kernel matrix is still needed for the deflation step. The main inconvenience of rKOPLS in relation to KPLS2 it that it requires the inversion of a matrix of size $R \times R$. However, this normally

|  | KOPLS | rKOPLS | KPLS2 |
|---|---|---|---|
| Number of nodes | $l$ | $R$ | $l$ |
| Size of Kernel Matrix | $l \times l$ | $R \times l$ | $l \times l$ |
| Storage requirements | $O(l^2)$ | $O(R^2)$ | $O(l^2)$ |
| Maximum $n_p$ | $\leq \min\{r(\mathbf{\Phi}), r(\mathbf{Y})\}$ | $\leq \min\{R, r(\mathbf{\Phi}), r(\mathbf{Y})\}$ | $\leq r(\mathbf{\Phi})$ |

Table 1: Summary of the most relevant characteristics of the proposed KOPLS and rKOPLS algorithms. Complexity for KPLS2 is also included for comparison purposes. We denote the rank of a matrix with $r(\cdot)$.

|  | # Train/Test | # Clases | dim | $\nu$-SVM (%) (linear) |
|---|---|---|---|---|
| *vehicle* | 500 / 346 | 4 | 18 | 66.18 |
| *segmentation* | 1310 / 1000 | 7 | 18 | 91.7 |
| *optdigits* | 3823 / 1797 | 10 | 64 | 96.33 |
| *satellite* | 4435 / 2000 | 6 | 36 | 83.25 |
| *pendigits* | 7494 / 3498 | 10 | 16 | 94.77 |
| *letter* | 10000 / 10000 | 26 | 16 | 79.81 |

Table 2: UCI benchmark datasets. Accuracy error rates for a linear $\nu$-SVM are also provided.

pays off in terms of reduction of computational time and storage requirements. In addition to this, our extensive simulation work shows that the projections provided by rKOPLS are generally more relevant than those of KPLS2.

## 4   Experiments

In this section, we will illustrate the ability of rKOPLS to discover relevant projections of the data. To do this, we compare the discriminative power of the features extracted by rKOPLS and KPLS2 in several multi-class classification problems. In particular, we include experiments on a benchmark of problems taken from the repository at the University of California Irvine (UCI) [2], and on a musical genre classification problem. This latter task is a good example of an application where rKOPLS can be specially useful, given the fact that the extraction of features from the raw audio data normally results in very large data sets of high dimensional data.

### 4.1   UCI Benchmark Data Sets

We start by analyzing the performance of our method in six standard UCI multi-class classification problems. Table 2 summarizes the main properties of the problems that constitute our benchmark. The last four problems can be considered large problems for MVA algorithms, which are in general not sparse and require the computation of the kernels between any two points in the training set. Our first set of experiments consists on comparing the discriminative performance of the features calculated by rKOPLS and KPLS2. For classification, we use one of the simplest possible models: we compute the pseudoinverse of the projected training data to calculate $\hat{\mathbf{B}}$ (see Eq. (3)), and then classify according to $\tilde{\mathbf{\Phi}}' \hat{\mathbf{B}}$ using a "winner-takes-all" (w.t.a.) activation function. For the kernel MVA algorithms we used a Gaussian kernel

$$k(\mathbf{x}_i, \mathbf{x}_j) = \exp\left(-\|\mathbf{x}_i - \mathbf{x}_j\|_2^2 / 2\sigma^2\right)$$

using 10-fold cross-validation (10-CV) on the training set to estimate $\sigma$. To obtain some reference accuracy rates, we also trained a $\nu$-SVM with Gaussian kernel, using the LIBSVM implementation[3] and 10-CV was carried out for both the kernel width and $\nu$.

Accuracy error rates for rKOPLS and different values of $R$ are displayed in the first rows and first columns of Table 3. Comparing these results with SVM (under the rbf-SVM column), we can

| | rKOPLS - pseudo+w.t.a. | | | KPLS2 - pseudo + w.t.a | | | |
|---|---|---|---|---|---|---|---|
| | $R = 250$ | $R = 500$ | $R = 1000$ | $l' = \sqrt{250\,l}$ | $l' = \sqrt{500\,l}$ | $l' = \sqrt{1000\,l}$ | $l' = l$ |
| *vehicle* | $80.4 \pm 1.2$ | $79.9$ | — | $\mathbf{81.3 \pm 1.3}$ | $80.5$ | — | $80.5$ |
| *segmentation* | $\mathbf{95.7 \pm 0.4}$ | $95.5 \pm 0.3$ | — | $93.9 \pm 0.5$ | $94.2 \pm 0.5$ | — | $95.1$ |
| *optdigits* | $97.4 \pm 0.2$ | $97.7 \pm 0.1$ | $\mathbf{98.2 \pm 0.2}$ | $96.5 \pm 0.3$ | $97 \pm 0.3$ | $97 \pm 0.2$ | $97.6$ |
| *satellite* | $89.8 \pm 0.2$ | $90.6 \pm 0.3$ | $91 \pm 0.2$ | $89.7 \pm 0.4$ | $90.3 \pm 0.6$ | $91.1 \pm 0.3$ | $\mathbf{91.8}$ |
| *pendigits* | $97.6 \pm 0.1$ | $\mathbf{98.2 \pm 0.1}$ | $98.1 \pm 0.2$ | $97.4 \pm 0.2$ | $97.6 \pm 0.1$ | $97.7 \pm 0.2$ | $96.9$ |
| *letter* | $84.8 \pm 0.3$ | $90 \pm 0.2$ | $\mathbf{92.9 \pm 0.4}$ | $84 \pm 0.6$ | $86 \pm 0.6$ | $86.2 \pm 0.4$ | — |
| | rKOPLS - SVM | | | KPLS2 - SVM | | | rbf-SVM |
| *vehicle* | $\mathbf{81.2 \pm 1}$ | $80.3$ | — | $\mathbf{81.2 \pm 1.1}$ | $80.6$ | — | $83$ |
| *segmentation* | $95.1 \pm 2$ | $95.4 \pm 0.4$ | — | $\mathbf{95.6 \pm 0.5}$ | $94.8 \pm 0.3$ | — | $95.2$ |
| *optdigits* | $97.3 \pm 0.2$ | $97.6 \pm 0.1$ | $\mathbf{98.2 \pm 0.2}$ | $96.4 \pm 0.2$ | $96.9 \pm 0.2$ | $96.9 \pm 0.3$ | $97.2$ |
| *satellite* | $89.6 \pm 0.6$ | $90.5 \pm 0.4$ | $\mathbf{91 \pm 0.2}$ | $89.7 \pm 0.5$ | $90.4 \pm 0.6$ | $90.8 \pm 0.5$ | $91.9$ |
| *pendigits* | $97.6 \pm 0.2$ | $\mathbf{98.2 \pm 0.1}$ | $98.1 \pm 0.2$ | $96.9 \pm 0.1$ | $97.1 \pm 0.2$ | $97.3 \pm 0.2$ | $98.1$ |
| *letter* | $88.8 \pm 1.5$ | $92.1 \pm 0.2$ | $\mathbf{93.9 \pm 0.3}$ | $85.8 \pm 0.5$ | $85.9 \pm 1.1$ | $87.7 \pm 1.2$ | $96.2$ |

Table 3: Classification performance in a benchmark of UCI datasets. Accuracy rates (%) and standard deviation of the estimation are given for 10 different runs of rKOPLS and KPLS2, both when using the pseudoinverse of the projected data together with the "winner-takes-all" activation function (first rows), and when using a $\nu$-SVM linear classifier (last rows). The results achieved by an SVM with linear classifier are also provided in the bottom right corner.

conclude that the rKOPLS approach is very close in performance or better than SVM in four out of the six problems. A clearly worse performance is observed in the smallest data set (*vehicle*) due to overfitting. For *letter*, we can see that, even for $R = 1000$, accuracy rates are far from those of SVM. The reason for this is that SVM is using 6226 support vectors, so that a very dense architecture seems to be necessary for this particular problem.

To make a fair comparison with the KPLS2 method, the training dataset was subsampled, selecting at random $l'$ samples, with $l'$ being the first integer larger than or equal to $\sqrt{R \times l}$. In this way, both rKOPLS and KPLS2 need the same number of kernel evaluations. Note that, even in this case, KPLS2 results in an architecture with $l'$ nodes ($l' > R$), so that projections of data are more expensive than for the respective rKOPLS. In any case, we must point out that subsampling was only considered for training the projections, but all training data was used to compute the pseudoinverse of the projected training data. Results without subsampling are also provided in Table 3 under the $l' = l$ column except for the *letter* data set which we were unable to process due to massive memory problems.

As a first comment, we have to point out that all the results for KPLS2 were obtained using 100 projections, which were necessary to guarantee the convergence of the method. In contrast to this, the maximum number of projections that the rKOPLS can provide equals the rank of the label matrix, i.e., the number of classes of each problem minus 1. In spite of using a much smaller number of projections, our algorithm performed significantly better than KPLS2 with subsampling in four out of the five largest problems.

As a final set of experiments, we have replaced the classification step by a linear $\nu$-SVM. The results, which are displayed in the bottom part of Table 3, are in general similar to those obtained with the pseudoinverse approach, both for rKOPLS and KPLS2. However, we can see that the linear SVM is able to better exploit the projections provided by the MVA methods in *vehicle* and *letter*, precisely the two problems where previous results were less satisfactory.

Based on the above set of experiments, we can conclude that rKOPLS provides more discriminative features than KPLS2. In addition to this, these projections are more "informative", in the sense that we can obtain a better recognition accuracy using a smaller number of projections. An additional advantage of rKOPLS in relation to KPLS2 is that it provides architectures with less nodes.

## 4.2 Feature Extraction for Music Genre Classification

In this subsection we consider the problem of predicting the genre of a song using the audio data only, a task which since the seminal paper [14] has been subject of much interest. The data set we

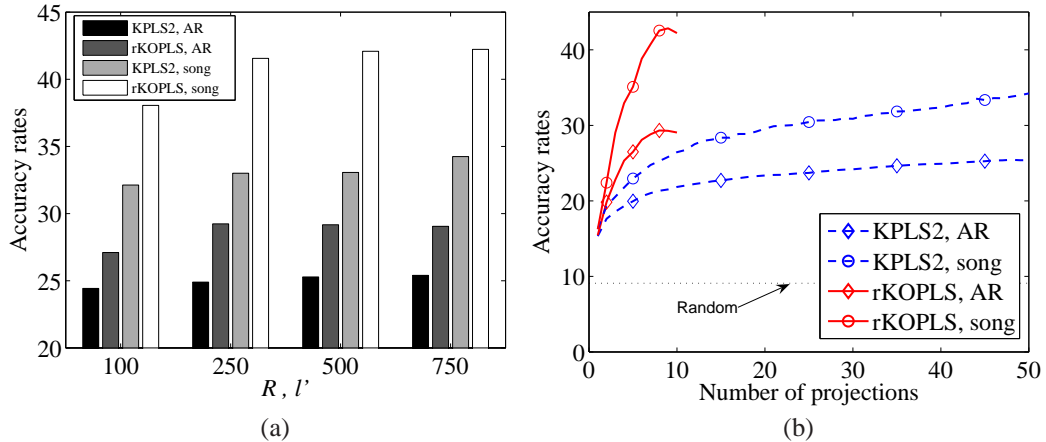

Figure 1: Genre classification performance of KPLS2 and rKOPLS.

analyze has been previously investigated in [5], and consists of 1317 snippets each of 30 seconds distributed evenly among 11 music genres: alternative, country, easy listening, electronica, jazz, latin, pop&dance, rap&hip-hop, r&b, reggae and rock. The music snippets are MP3 (MPEG1-layer3) encoded music with a bitrate of 128 kbps or higher, down sampled to 22050 Hz, and they are processed following the method in [5]: MFCC features are extracted from overlapping frames of the song, using a window size of 20 ms. Then, to capture temporal correlation, a Multivariate Autoregressive (AR) model is adjusted for every 1.2 seconds of the song, and finally the parameters of the AR model are stacked into a 135 length feature vector for every such frame.

For training and testing the system we have split the data set into two subsets with 817 and 500 songs, respectively. After processing the audio data, we have 57388 and 36556 135-dimensional vectors in the training and test partitions, an amount which for most kernel MVA methods is prohibitively large. For the rKOPLS, however, the compact representation is enabling usage of the entire training data.

In Figure 1 the results are shown. Note that, in this case, comparisons between rKOPLS and KPLS2 are for a fixed architecture complexity ($R = l'$), since the most significant computational burden for the training of the system is in the projection of the data. Since every song consists of about seventy AR vectors, we can measure the classification accuracy in two different ways: 1) On the level of individual AR vectors or 2) by majority voting among the AR vectors of a given song. The results shown in Figure 1 are very clear: Compared to KPLS2, the rKOPLS is not only consistently performing better as seen in Figure 1(a), but is also doing so with much fewer projections. The strong results are very pronounced in Figure 1(b) where, for $R = 750$, rKOPLS is outperforming ordinary KPLS, and is doing so with only ten projections compared to fifty projections of the KPLS2. This demonstrates that the features extracted by rKOPLS holds much more information relevant to the genre classification task than KPLS2.

## 5  Conclusions

In this paper we have presented a novel kernel PLS algorithm, that we call reduced kernel orthonormalized PLS (rKOPLS). Compared to similar approaches, rKOPLS is making the data in feature space orthonormal, and imposing sparsity on the solution to ensure competitive performance on large data sets.

Our method has been tested on a benchmark of UCI data sets, and we have found that the results were competitive in comparison to those of rbf-SVM, and superior to those of the ordinary KPLS2 method. Furthermore, when applied to a music genre classification task, rKOPLS performed very well even with only a few features, keeping also the complexity of the algorithm under control.

Because of the nature of music data, in which both the number of dimensions and samples are very large, we believe that feature extraction methods such as rKOPLS can become crucial to music information retrieval tasks, and hope that other researchers in the community will be able to benefit from our results.

**Acknowledgments**

This work was partly supported by the Danish Technical Research Council, through the framework project 'Intelligent Sound', www.intelligentsound.org (STVF No. 26-04-0092), and by the Spanish Ministry of Education and Science with a Postdoctoral Felowship to the first author.

## Footnotes

[1]Centering of data in feature space can easily be done from the original kernel matrix. Details on this process are given in most text books describing kernel methods, e.g. [13, 12].

[2]http://www.ics.uci.edu/~mlearn/MLRepository.html

[3]Software available at http://www.csie.ntu.edu.tw/~cjlin/libsvm

# References

[1] Paul Geladi. Notes on the history and nature of partial least squares (PLS) modelling. *Journal of Chemometrics*, 2:231–246, 1988.

[2] L. Hoegaerts, J. A. K. Suykens, J. Vanderwalle, and B. De Moor. Primal space sparse kernel partial least squares regression for large problems. In *Proceedings of International Joint Conference on Neural Networks (IJCNN)*, 2004.

[3] Agnar Hoskuldsson. PLS regression methods. *Journal of Chemometrics*, 2:211–228, 1988.

[4] Yuh-Jye Lee and O. L. Mangasarian. RSVM: reduced support vector machines. In *Data Mining Institute Technical Report 00-07, July 2000. CD Proceedings of the SIAM International Conference on Data Mining, Chicago, April 5-7, 2001,*, 2001.

[5] Anders Meng, Peter Ahrendt, Jan Larsen, and Lars Kai Hansen. Temporal feature integration for music genre classification. *IEEE Trans. Audio, Speech & Language Process.*, to appear.

[6] Michinari Momma and Kristin Bennett. Sparse kernel partial least squares regression. In *Proceedings of Conference on learning theory (COLT)*, 2003.

[7] Roman Rosipal and Leonard J. Trejo. Kernel partial least squares regression in reproducing kernel hilbert space. *Journal of Machine Learning Research*, 2:97–123, 2001.

[8] Roman Rosipal, Leonard J. Trejo, and Bryan Matthews. Kernel pls-svc for linear and nonlinear classifiction. In *Proceedings of Internation Conference on Machine Learning (ICML)*, 2003.

[9] Kramer N. Rosipal R. Overview and recent advances in partial least squares. In *Subspace, Latent Structure and Feature Selection Techniques*, 2006.

[10] Sam Roweis and Carlos Brody. Linear heteroencoders. Technical report, Gatsby Computational Neuroscience Unit, 1999.

[11] Paul D. Sampson, Ann P. Streissguth, Helen M. Barr, and Fred L. Bookstein. Neurobehavioral effetcs of prenatal alcohol: Part II. Partial Least Squares analysis. *Neurotoxicology and teratology*, 11:477–491, 1989.

[12] Bernhard Schoelkopf and Alexander Smola. *Learning with kernels*. MIT Press, 2002.

[13] John Shawe-Taylor and Nello Christiani. *Kernel Methods for Pattern Analysis*. Cambridge University Press, 2004.

[14] George Tzanetakis and Perry Cook. Music genre classification of audio signals. *IEEE Transactions on Speech and Audio Processing*, 10(5):293–302, July 2002.

[15] Jacob A. Wegelin. A survey of partial least squares (PLS) methods, with emphasis on the two-block case. Technical report, University of Washington, 2000.

[16] Herman Wold. Path models with latent variables: the NIPALS approach. In *Quatitative sociology: International perspectives on mathematical and statistical Model Building*, pages 307–357. Academic Press, 1975.

[17] S. Wold, C. Albano, W. J. Dunn, U. Edlund, K. Esbensen, P. Geladi, S. Hellberg, E. Johansson, W. Lindberg, and M. Sjostrom. *Chemometrics, Mathematics and Statistics in Chemistry*, chapter Multivariate Data Analysis in Chemistry, page 17. Reidel Publishing Company, 1984.

[18] K. Worsley, J. Poline, K. Friston, and A. Evans. Characterizing the response of pet and fMRI data using multivariate linear models (MLM). *NeuroImage*, 6:305– 319, 1998.
